# Learning Preferences for Multiclass Problems

**Fabio Aiolli**
Dept. of Computer Science
University of Pisa, Italy
aiolli@di.unipi.it

**Alessandro Sperduti**
Dept. of Pure and Applied Mathematics
University of Padova, Italy
sperduti@math.unipd.it

## Abstract

Many interesting multiclass problems can be cast in the general framework of label ranking defined on a given set of classes. The evaluation for such a ranking is generally given in terms of the number of violated order constraints between classes. In this paper, we propose the *Preference Learning Model* as a unifying framework to model and solve a large class of multiclass problems in a large margin perspective. In addition, an original kernel-based method is proposed and evaluated on a ranking dataset with state-of-the-art results.

## 1 Introduction

The presence of multiple classes in a learning domain introduces interesting tasks besides the one to select the most appropriate class for an object, the well-known (*single-label*) multiclass problem. Many others, including learning rankings, multi-label classification, hierarchical classification and ordinal regression, just to name a few, have not yet been sufficiently studied even though they should not be considered less important. One of the major problems when dealing with this large set of different settings is the lack of a single universal theory encompassing all of them.

In this paper we focus on multiclass problems where labels are given as partial order constraints over the classes. Tasks naturally falling into this family include *category ranking*, which is the task to infer full orders over the classes, *binary category ranking*, which is the task to infer orders such that a given subset of classes are top-ranked, and any general (q-label) classification problem.

Recently, efforts have been made in the direction to unify different ranking problems. In particular, in [5, 7] two frameworks have been proposed which aim at inducing a label ranking function from examples. Similarly, here we consider labels coded into sets of preference constraints, expressed as *preference graphs* over the set of classes. The multiclass problem is then reduced to learning a good set of *scoring functions* able to correctly rank the classes according to the constraints which are associated to the label of the examples. Each preference graph disagreeing with the obtained ranking function will count as an error.

The primary contribution of this work is to try to make a further step towards the unification of different multiclass settings, and different models to solve them, by proposing the *Preference Learning Model*, a very general framework to model and study several kinds of multiclass problems. In addition, a kernel-based method particularly suited for this setting is proposed and evaluated in a binary category ranking task with very promising results.

**The Multiclass Setting**   Let $\Omega$ be a set of classes, we consider a multiclass setting where data are supposed to be sampled according to a probability distribution $\mathcal{D}$ over $\mathcal{X} \times \mathcal{Y}$, $\mathcal{X} \subseteq \mathbb{R}^d$ and an hypothesis space of functions $\mathcal{F} = \{f_\Theta : \mathcal{X} \times \Omega \to \mathbb{R}\}$ with parameters $\Theta$. Moreover, a cost function $c(\mathbf{x}, y|\Theta)$ defines the cost suffered by a given hypothesis on a pattern $\mathbf{x} \in \mathcal{X}$ having label $y \in \mathcal{Y}$. A multiclass learning algorithm searches for a set of parameters $\Theta^*$ such to minimize the *true cost*, that is the expected value of the cost according to the true distribution of data, i.e. $R_t[\Theta] = E_{(\mathbf{x},y)\sim\mathcal{D}}[c(\mathbf{x}, y|\Theta)]$. The distribution $\mathcal{D}$ is typically unknown, while it is available a training set $\mathcal{S} = \{(\mathbf{x}_1, y_1), \ldots, (\mathbf{x}_n, y_n)\}$ with examples drawn $i.i.d.$ from $\mathcal{D}$. An empirical approximation of the true cost, also referred to as the *empirical cost*, is defined by $R_e[\Theta, \mathcal{S}] = \frac{1}{n} \sum_{i=1}^n c(\mathbf{x}_i, y_i|\Theta)$.

## 2   The Preference Learning Model

In this section, starting from the general multiclass setting described above, we propose a general technique to solve a large family of multiclass settings. The basic idea is to "code" labels of the original multiclass problem as sets of ranking constraints given as preference graphs. Then, we introduce the *Preference Learning Model* (PLM) for the induction of optimal scoring functions that uses those constraints as supervision.

In the case of ranking-based multiclass settings, labels are given as partial orders over the classes (see [1] for a detailed taxonomy of multiclass learning problems). Moreover, as observed in [5], ranking problems can be generalized by considering labels given as preference graphs over a set of classes $\Omega = \{\omega_1, \ldots, \omega_m\}$, and trying to find a consistent ranking function $f_R : \mathcal{X} \to \Pi(\Omega)$ where $\Pi(\Omega)$ is the set of permutations over $\Omega$. More formally, considering a set $\Omega$, a *preference graph* or *"p-graph"* over $\Omega$ is a directed graph $v = (N, A)$ where $N \subseteq \Omega$ is the set of nodes and $A$ is the set of arcs of the graph accessed by the function $A(v)$. An arc $a \in A$ is associated with its starting node $\omega_s = \omega_s(a)$ and its ending node $\omega_e = \omega_e(a)$ and represents the information that the class $\omega_s$ is preferred to, and should be ranked higher than, $\omega_e$. The set of p-graphs over $\Omega$ will be denoted by $G(\Omega)$.

Let be given a set of scoring functions $f : \mathcal{X} \times \Omega \to \mathbb{R}$ with parameters $\Theta$ working as predictors of the relevance of the associated class to given instances. A definition of a ranking function naturally follows by taking the permutation of elements in $\Omega$ corresponding to the sorting of the values of these functions, i.e. $f_R(\mathbf{x}|\Theta) = \text{argsort}_{\omega \in \Omega} f(\mathbf{x}, \omega|\Theta)$. We say that a preference arc $a = (\omega_s, \omega_e)$ is consistent with a ranking hypothesis $f_R(\mathbf{x}|\Theta)$, and we write $a \sqsubseteq f_R(\mathbf{x}|\Theta)$, when $f(\mathbf{x}, \omega_s|\Theta) \geq f(\mathbf{x}, \omega_e|\Theta)$ holds. Generalizing to graphs, a p-graph $g$ is said to be consistent with an hypothesis $f_R(\mathbf{x}|\Theta)$, and we write $g \sqsubseteq f_R(\mathbf{x}|\Theta)$, if every arc compounding it is consistent, i.e. $g \sqsubseteq f_R(\mathbf{x}|\Theta) \Leftrightarrow \forall a \in A(g), a \sqsubseteq f_R(\mathbf{x}|\Theta)$.

**The PLM Mapping**   Let us start by considering the way a multiclass problem is transformed into a PLM problem. As seen before, to evaluate the quality of a ranking function $f_R(\mathbf{x}|\Theta)$ is necessary to specify the nature of a cost function $c(\mathbf{x}, y|\Theta)$. Specifically, we consider cost definitions corresponding to associate penalties whenever uncorrect decisions are made (e.g. a classification error for classification problems or wrong ordering for ranking problems). To this end, as in [5], we consider a label mapping $\mathcal{G} : y \mapsto \{g_1(y), \ldots, g_{q_y}(y)\}$ where a set of subgraphs $g_i(y) \in G(\Omega)$ are associated to each label $y \in \mathcal{Y}$. The total cost suffered by a ranking hypothesis $f_R$ on the example $\mathbf{x} \in \mathcal{X}$ labeled $y \in \mathcal{Y}$ is the number of p-graphs in $\mathcal{G}(y)$ not consistent with the ranking, i.e. $c(\mathbf{x}, y|\Theta) = \sum_{j=1}^{q_y} [\![g_j(y) \not\sqsubseteq f(\mathbf{x}|\Theta)]\!]$, where $[\![b]\!]$ is 1 if the condition $b$ holds, 0 otherwise. Let us describe three particular mappings proposed in [5] that seem worthwhile of note: *(i)* The *identity mapping*, denoted by $\mathcal{G}_I$, where the label is mapped on itself and every inconsistent graph will have a unitary cost, *(ii)* the *disagreement mapping*, denoted by $\mathcal{G}_d$, where a simple (single-preference) subgraph is built for each arc in $A(y)$, and *(iii)* the *domination mapping*, denoted by $\mathcal{G}_D$, where for each node $\omega_r$ in $y$ a subgraph consisting of $\omega_r$ plus

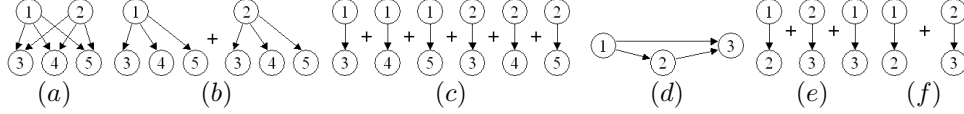

Figure 1: Examples of label mappings for 2-label classification (a-c) and ranking (d-f).

the nodes of its outgoing set is built. To clarify, in Figure 1 a set of mapping examples are proposed. Considering $\Omega = \{1, 2, 3, 4, 5\}$, in Figure 1-(a) the label $y = [1, 2|3, 4, 5]$ for a 2-label classification setting is given. In particular, this corresponds to the mapping $\mathcal{G}(y) = \mathcal{G}_I(y) = y$ where a single wrong ranking of a class makes the predictor to pay a unit of cost. Similarly, in Figure 1-(b) the label mapping $\mathcal{G}(y) = \mathcal{G}_D(y)$ is presented for the same problem. Another variant is presented in Figure 1-(c) where the label mapping $\mathcal{G}(y) = \mathcal{G}_d(y)$ is used and the target classes are independently evaluated and their errors cumulated. Note that all these graphs are subgraphs of the original label in 1-(a). As an additional example we consider the three cases depicted in the right hand side of Figure 1 that refer to a ranking problem with three classes $\Omega = \{1, 2, 3\}$. In Figure 1-(d) the label $y = [1|2|3]$ is given. As before, this also corresponds to the label mapping $\mathcal{G}(y) = \mathcal{G}_I(y)$. Two alternative cost definitions can be obtained by using the p-graphs (sets of basic preferences actually) depicted in Figure 1-(e) and 1-(f). Note that the cost functions in these cases are different. For example, assume $f_R(\mathbf{x}|\Theta) = [3|1|2]$, the p-graph in (e) induces a cost $c(\mathbf{x}, y_b|\Theta) = 2$ while the p-graph in (f) induces a cost $c(\mathbf{x}, y_c|\Theta) = 1$.

**The PLM Setting** Once the label mapping $\mathcal{G}$ is fixed, the preference constraints of the original multiclass problem can be arranged into a set of preference constraints. Specifically, we consider the set $\mathcal{V}(\mathcal{S}) = \bigcup_{(\mathbf{x}_i, y_i) \in \mathcal{S}} \mathcal{V}(\mathbf{x}_i, y_i)$ where $\mathcal{V}(\mathbf{x}, y) = \{(\mathbf{x}, g_j(y))\}_{j \in \{1, .., q_y\}}$ and each pair $(\mathbf{x}, g) \in \mathcal{X} \times G(\Omega)$ is a preference constraint. Note that the same instance can be replicated in $\mathcal{V}(\mathcal{S})$. This can happen, for example, when multiple ranking constraints are associated to the same example of the original multiclass problem. Because of this, in the following, we prefer to use a different notation for the instances in preference constraints to avoid confusion with training examples.

Notions defined for the standard classification setting are easily extended to PLM. For a preference constraint $(\mathbf{v}, g) \in \mathcal{V}$, the *constraint error* incurred by the ranking hypothesis $f_R(\mathbf{v}|\Theta)$ is given by $\delta(\mathbf{v}, g|\Theta) = [\![ g \not\sqsubseteq f_R(\mathbf{v}|\Theta) ]\!]$. The empirical cost is then defined as the cost over the whole constraint set, i.e. $R_e[\Theta, \mathcal{V}] = \sum_{i=1}^{N} \delta(\mathbf{v}_i, g_i|\Theta)$. In addition, we define the *margin* of an hypothesis on a pattern $\mathbf{v}$ for a preference arc $a = (\omega_s, \omega_e)$, expressing how well the preference is satisfied, as the difference between the scores of the two linked nodes, i.e. $\rho_A(\mathbf{v}, a|\Theta) = f(\mathbf{v}, \omega_s|\Theta) - f(\mathbf{v}, \omega_e|\Theta)$. The margin for a p-graph constraint $(\mathbf{v}, g)$ is then defined as the minimum of the margin of the compounding preferences, $\rho_G(\mathbf{v}, g|\Theta) = \min_{a \in A(g)} \rho_A(\mathbf{v}, a|\Theta)$, and gives a measure of how well the hypothesis fulfills a given preference constraint. Note that, consistently with the classification setting, the margin is greater than 0 if and only if $g \sqsubseteq f_R(\mathbf{v}|\Theta)$.

**Learning in PLM** In the PLM we try to learn a "simple" hypothesis able to minimize the empirical cost of the original multiclass problem or equivalently to satisfy the constraints in $\mathcal{V}(\mathcal{S})$ as much as possible. The learning setting of the PLM can be reduced to the following scheme. Given a set $\mathcal{V}$ of pairs $(\mathbf{v}_i, g_i) \in \mathcal{X} \times G(\Omega)$, $i \in \{1, \ldots, N\}$, $N = \sum_{i=1}^{n} q_{y_i}$, find a set of parameters for the ranking function $f_R(\mathbf{v}|\Theta)$ able to minimize a combination of a regularization and an empirical loss term, $\hat{\Theta} = \arg \min_{\Theta} \{ R_e[\Theta, \mathcal{V}] + \mu \mathcal{R}(\Theta) \}$ with $\mu$ a given constant. However, since the direct minimization of this functional is hard due to the non continuous form of the empirical error term, we use an upper-bound on the true empirical error. To this end, let be defined a monotonically decreasing loss function $L$ such

that $L(\rho) \geq 0$ and $L(0) = 1$, then by defining a margin-based loss

$$L_C(\mathbf{v}, g|\Theta) = L\left(\rho_G(\mathbf{v}, g|\Theta)\right) = \max_{a \in A(g)} L\left(\rho_A(\mathbf{v}, a|\Theta)\right) \tag{1}$$

for a p-graph constraint $(\mathbf{v}, g) \in \mathcal{V}$ and recalling the margin definition, the condition $\delta(\mathbf{v}, g|\Theta) \leq L_C(\mathbf{v}, g|\Theta)$ always holds thus obtaining $R_e[\Theta, \mathcal{V}] \leq \sum_{i=1}^{N} L_C(\mathbf{v}_i, g_i|\Theta)$.

The problem of learning with multiple classes (up to constant factors) is then reduced to a minimization of a (possibly regularized) loss functional

$$\hat{\Theta} = \arg \min_{\Theta} \{\mathcal{L}(\mathcal{V}|\Theta) + \mu \mathcal{R}(\Theta)\} \tag{2}$$

where $\mathcal{L}(\mathcal{V}|\Theta) = \sum_{i=1}^{N} \max_{a \in A(g_i)} L(f(\mathbf{v}_i, \omega_s(a)|\Theta) - f(\mathbf{v}_i, \omega_e(a)|\Theta))$.

| Method | $L(\rho)$ |
|--------|-----------|
| $\beta$-margin Perceptron | $[1 - \beta^{-1}\rho]_+$ |
| Logistic Regression | $\log_2(1 + \exp(-\rho))$ |
| Soft margin | $[1 - \rho]_+$ |
| Mod. Least Square | $[1 - \rho]_+^2$ |
| Exponential | $\exp(-\rho)$ |

Many different choices can be made for the function $L(\cdot)$. Some well known examples are the ones given in the table at the left. Note that, if the function $L(\cdot)$ is convex with respect to the parameters $\Theta$, the minimization of the functional in Eq. (2) will result quite easy given a convex regularization term. The only difficulty in this case is represented by the $max$ term. A shortcoming to this problem would consist in upper-bounding the $max$ with the $sum$ operator, though this would probably lead to a quite row approximation of the indicator function when considering p-graphs with many arcs. It can be shown that a number of related works, e.g. [5, 7], after minor modifications, can be seen as PLM instances when using the $sum$ approximation. Interestingly, PLM highlights that this approximation in fact corresponds to a change on the label mapping obtained by decomposing a complex preference graph into a set of binary preferences and thus changing the cost definition we are indeed minimizing. In this case, using either $\mathcal{G}_D$ or $\mathcal{G}_d$ is not going to make any difference at all.

**Multiclass Prediction through PLM** A multiclass prediction is a function $H : \mathcal{X} \to \mathcal{Y}$ mapping instances to their associated label. Let be given a label mapping defined as $\mathcal{G}(y) = \{g_1(y), \dots, g_{q_y}(y)\}$. Then, the PLM multiclass prediction is given as the label whose induced preference constraints mostly agree with the current hypothesis, i.e. $H(\mathbf{x}) = \arg \min_y \mathcal{L}(\mathcal{V}(\mathbf{x}, y)|\Theta)$ where $\mathcal{V}(\mathbf{x}, y) = \{(\mathbf{x}, g_j(y))\}_{j \in \{1,..,q_y\}}$. It can be shown that many of the most effective methods used for learning with multiple classes, including output coding (ECOC, OvA, OvO), boosting, least squares methods and all the methods in [10, 3, 7, 5] fit into the PLM setting. This issue is better discussed in [1].

## 3 Preference Learning with Kernel Machines

In this section, we focus on a particular setting of the PLM framework consisting of a multivariate embedding $\mathbf{h} : \mathcal{X} \to \mathbb{R}^s$ of linear functions parameterized by a set of vectors $W_k \in \mathbb{R}^d$, $k \in \{1, \dots, s\}$ accommodated in a matrix $W \in \mathbb{R}^{s \times d}$, i.e. $\mathbf{h}(\mathbf{x}) = [h_1(\mathbf{x}), \dots, h_s(\mathbf{x})] = [\langle W_1, \mathbf{x} \rangle, \dots, \langle W_s, \mathbf{x} \rangle]$. Furthermore, we consider the set of classes $\Omega = \{\omega_1, \dots, \omega_m\}$ and $M \in \mathbb{R}^{m \times s}$ a matrix of codes of length $s$ with as many rows as classes. This matrix has the same role as the coding matrix in multiclass coding, e.g. in ECOC. Finally, the scoring function for a given class is computed as the dot product between the embedding function and the class code vector

$$f(\mathbf{x}, \omega_r|W, M) = \langle \mathbf{h}(\mathbf{x}), M_r \rangle = \sum_{k=1}^{s} M_{rk} \langle W_k, \mathbf{x} \rangle \tag{3}$$

Now, we are able to describe a kernel-based method for the effective solution of the PLM problem. In particular, we present the problem formulation and the associated optimization method for the task of learning the embedding function given fixed codes for the classes (*embedding* problem). Another worthwhile task consists in the optimization of the codes for the classes when the embedding function is kept fixed (*coding* problem), or even to perform a combination of the two (see for example [8]). A deeper study of the embedding-coding version of PLM and a set of examples can be found in [1].

**PLM Kesler's Construction**    As a first step, we generalize the Kesler's Construction originally defined for single-label classification (see [6]) to the PLM setting, thus showing that the embedding problem can be formulated as a binary classification problem in a higher dimensional space when new variables are appropriately defined. Specifically, consider the vector $\mathbf{y}(a) = (M_{\omega_s(a)} - M_{\omega_e(a)}) \in \mathbb{R}^s$ defined for every preference arc in a given preference constraint, that is $a = (\omega_s, \omega_e) \in A(g)$. For every instance $\mathbf{v}_i$ and preference $(\omega_s, \omega_e)$, the preference condition $\rho_A(\mathbf{v}_i, a) \geq 0$ can be rewritten as

$$
\begin{aligned}
\rho_A(\mathbf{v}_i, a) &= f(\mathbf{v}_i, \omega_s) - f(\mathbf{v}_i, \omega_e) &= \langle \mathbf{y}(a), \mathbf{h}(\mathbf{v}_i) \rangle &= \sum_{k=1}^s y_k(a) \langle W_k, \mathbf{v}_i \rangle \\
&= \sum_{k=1}^s \langle W_k, y_k(a)\mathbf{v}_i \rangle &= \sum_{k=1}^s \langle W_k, [\mathbf{z}_i^a]_k^s \rangle &= \langle \mathbf{W}, \mathbf{z}_i^a \rangle \geq 0
\end{aligned}
\tag{4}
$$

where $[\cdot]_k^s$ denotes the $k$-th chunk of a s-chunks vector, $\mathbf{W} \in \mathbb{R}^{s \cdot d}$ is the vector obtained by sequentially arranging the vectors $W_k$, and $\mathbf{z}_i^a = \mathbf{y}(a) \otimes \mathbf{v}_i \in \mathbb{R}^{s \cdot d}$ is the embedded vector made of the $s$ chunks defined by $[\mathbf{z}_i^a]_k^s = y_k(a)\mathbf{v}_i$, $k \in \{1, \ldots, s\}$. From this derivation it turns out that each preference of a constraint in the set $\mathcal{V}$ can be viewed as an example of dimension $s \cdot d$ in a binary classification problem. Each pair $(\mathbf{v}_i, g_i) \in \mathcal{V}$ then generates a number of examples in this extended binary problem equal to the number of arcs of the p-graph $g_i$ for a total of $\sum_{i=1}^N |A(g_i)|$ examples. In particular, the set $\mathcal{Z} = \{\mathbf{z}_i^a\}$ is linearly separable in the higher dimensional problem if and only if there exists a consistent solution for the original PLM problem. Very similar considerations, omitted for space reasons, could be given for the coding problem as well.

**The Kernel Preference Learning Optimization**    As pointed out before, the central task in PLM is to learn scoring functions in such a way to be as much as possible consistent with the set of constraints in $\mathcal{V}$. This is done by finding a set of parameters minimizing a loss function that is an upper-bound on the empirical error function. For the embedding problem, instantiating the problem (2), and choosing the 2-norm of the parameters as regularizer, we obtain $\hat{W} = \arg\min_W \frac{1}{N} \sum_{i=1}^N L_C(\mathbf{v}_i, g_i | W, M) + \mu ||W||^2$ where, according to Eq.(1), the loss for each preference constraint is computed as the maximum between the losses of all the associated preferences, that is $L_i = \max_{a \in A(g_i)} L(\langle \mathbf{W}, \mathbf{z}_i^a \rangle)$.

When the constraint set in $\mathcal{V}$ contains basic preferences only (that is p-graphs consisting of a single arc $a_i = A(g_i)$), the optimization problem can be simplified into the minimization of a standard functional combining a loss function with a regularization term. Specifically, all the losses presented before can be used and, for many of them, it is possible to give a kernel-based solution. See [11] for a set of examples of loss functions and the formulation of the associated problem with kernels.

**The Kernel Preference Learning Machine**    For the general case of p-graphs possibly containing multiple arcs, we propose a kernel-based method (hereafter referred to as *Kernel Preference Learning Machine* or KPLM for brevity) for PLM optimization which adopts the loss *max* in Eq. (2). Borrowing the idea of soft-margin [9], for each preference arc, a linear loss is used giving an upper bound on the indicator function loss. Specifically, we use the SVM-like soft margin loss $L(\rho) = [1 - \rho]_+$.

Summarizing, we require a set of small norm predictors that fulfill the soft constraints of

the problem. These requirements can be expressed by the following quadratic problem

$$\min_{\mathbf{W},\xi} \frac{1}{2}||\mathbf{W}||^2 + C\sum_i^N \xi_i$$
$$\text{subject to:} \begin{cases} \langle \mathbf{W}, \mathbf{z}_i^a \rangle \geq 1 - \xi_i, & i \in \{1,..,N\}, a \in A(g_i) \\ \xi_i \geq 0, & i \in \{1,..,N\} \end{cases} \tag{5}$$

Note that differently from the SVM formulation for the binary classification setting, here the slack variables $\xi_i$ are associated to multiple examples, one for each preference arc in the p-graph. Moreover, the optimal value of the $\xi_i$ corresponds to the loss value as defined by $L_i$. As it is easily verifiable, this problem is convex and it can be solved in the usual way by resorting to the optimization of the Wolfe dual problem. Specifically, we have to find the saddle point (minimization w.r.t. to the primal variables $\{\mathbf{W},\xi\}$ and maximization w.r.t. the dual variables $\{\alpha,\lambda\}$) of the following Lagrangian:

$$\mathcal{Q}(\mathbf{W},\xi,\alpha,\lambda) = \frac{1}{2}||\mathbf{W}||^2 + C\sum_i^N \xi_i + \sum_i^N \sum_{a\in A(g_i)} \alpha_i^a(1 - \xi_i - \langle \mathbf{W}, \mathbf{z}_i^a \rangle)$$
$$- \sum_i^N \lambda_i \xi_i, \text{ s.t. } \alpha_i^a, \lambda_i \geq 0 \tag{6}$$

By differentiating the Lagrangian with respect to the primal variables and imposing the optimality conditions we obtain the set of constraints that the variables have to fulfill in order to be an optimal solution

$$\frac{\partial \mathcal{Q}}{\partial \mathbf{W}} = \mathbf{W} - \sum_i^N \sum_{a\in A(g_i)} \alpha_i^a \mathbf{z}_i^a = 0 \Leftrightarrow \mathbf{W} = \sum_i^N \sum_{a\in A(g_i)} \alpha_i^a \mathbf{z}_i^a$$
$$\frac{\partial \mathcal{Q}}{\partial \xi_i} = C - \sum_{a\in A(g_i)} \alpha_i^a - \lambda_i = 0 \Leftrightarrow \sum_{a\in A(g_i)} \alpha_i^a \leq C \tag{7}$$

Substituting conditions (7) in (6) and omitting constants that do not change the solution, the problem can be restated as

$$\max_\alpha \sum_{i,a} \alpha_i^a - \frac{1}{2}\sum_k^s \sum_{i,a_i} \sum_{j,a_j} y_k(a_i)y_k(a_j)\alpha_i^{a_i}\alpha_j^{a_j}\langle \mathbf{v}_i, \mathbf{v}_j \rangle$$
$$\text{subject to:} \begin{cases} \alpha_i^a \geq 0, & i \in \{1,..,N\}, a \in A(g_i) \\ \sum_a \alpha_i^a \leq C, & i \in \{1,..,N\} \end{cases} \tag{8}$$

Since $W_k = \sum_{i,a} y_k(a)\alpha_i^a \mathbf{v}_i = \sum_{i,a}[M_{\omega_s(a)} - M_{\omega_e(a)}]_k^s \alpha_i^a \mathbf{v}_i$, $k = 1,..,s$, we obtain $h_k(\mathbf{x}) = \langle W_k, \mathbf{x} \rangle = \sum_{i,a}[M_{\omega_s(a)} - M_{\omega_e(a)}]_k^s \alpha_i^a \langle \mathbf{v}_i, \mathbf{x} \rangle$. Note that any kernel $k(\cdot,\cdot)$ can be substituted in place of the linear dot product $\langle,\rangle$ to allow for non-linear decision functions.

**Embedding Optimization** The problem in (8) recalls the one obtained for single-label multiclass SVM [1, 2] and, in fact, its optimization can be performed in a similar way. Assuming a number of arcs for each preference constraint equal to $q$, the dual problem in (8) involves $N \cdot q$ variables leading to a very large scale problem. However, it can be noted that the independence of constraints among the different preference constraints allows for the separation of the variables in $N$ disjoints sets of $q$ variables each.

The algorithm we propose for the optimization of the overall problem consists in iteratively selecting a preference constraint from the constraints set (a p-graph) and then optimizing with respect to the variables associated with it, that is one for each arc of the p-graph. From the convexity of the problem and the separation of the variables, since on each iteration we optimize on a different subset of variables, this guarantees that the optimal solution for the Lagrangian will be found when no new selections can lead to improvements.

The graph to optimize at each step is selected on the basis of an heuristic selection strategy. Let the preference constraint $(\mathbf{v}_i, g_i) \in \mathcal{V}$ be selected at a given iteration, to enforce the constraint $\sum_{a\in A(g_i)} \alpha_i^a + \lambda_i = C$, $\lambda_i \geq 0$, two elements from the set of variables $\{\alpha_i^a | a \in A(g_i)\} \cup \{\lambda_i\}$ will be optimized in pairs while keeping the solution inside the feasible region $\alpha_i^a \geq 0$. In particular, let $\chi_1$ and $\chi_2$ be the two selected variables, we restrict the

updates to the form $\chi_1 \leftarrow \chi_1 - \nu$ and $\chi_2 \leftarrow \chi_2 + \nu$ with optimal choices for $\nu$. The variables which most violate the constraints are iteratively selected until they reach optimality KKT conditions. For this, we have devised a KKT-based procedure which is able to select these variables in time linear with the number of classes. For space reasons we omit the details and we do not consider at all any implementation issue. Details and optimized versions of this basic algorithm can be found in [1].

**Generalization of KPLM**   As a first immediate result we can give an upper-bound on the leave-one-out error by utilizing the sparsity of a KPLM solution, namely $LOO \leq |V|/N$, where $V = \{i \in \{1, \ldots, N\} | \max_{a \in A(g_i)} \alpha_i^a > 0\}$ is the set of support vectors. Another interesting result about the generalization ability of a KPLM is in the following theorem.

**Theorem 1** *Consider a KPLM hypothesis* $\Theta = (W, M)$ *with* $\sum_{r=1}^s ||W_r||^2 = 1$ *and* $||M_r||^2 \leq R_M$ *such that* $\min_{(\mathbf{v},g) \in \mathcal{V}} \rho_G(\mathbf{v}, g|\Theta) \geq \gamma$. *Then, for any probability distribution* $\mathcal{D}$ *on* $\mathcal{X} \times \mathcal{Y}$ *with support in a ball of radius* $R_{\mathcal{X}}$ *around the origin, with probability* $1 - \delta$ *over* $n$ *random examples* $\mathcal{S}$, *the following bound for the true cost holds*

$$R_t[\Theta] \leq \frac{2QA}{n} \left( \frac{64R^2}{\gamma^2} \log \frac{en\gamma}{8R^2} \log \frac{32n}{\gamma^2} + \log \frac{4}{\delta} \right)$$

*where* $\forall y \in \mathcal{Y}$, $q_y \leq Q$, $|A(g_r(y))| \leq A$, $r \in \{1, \ldots, q_y\}$ *and* $R = 2R_M R_{\mathcal{X}}$.

*Proof.* Similar to that of Theorem 4.11 in [7] when noting that the size of examples in $\mathcal{Z}$ are upper-bounded by $R = 2R_M R_{\mathcal{X}}$.

## 4   Experiments

**Experimental Setting**   We performed experiments on the 'ModApte" split of Reuters-21578 dataset. We selected the 10 most popular categories thus obtaining a reduced set of 6,490 training documents and a set of 2,545 test documents. The corpus was then pre-processed by discarding numbers and punctuation and converting letters to lowercase. We used a stop-list to remove very frequent words and stemming has been performed by means of Porter's stemmer. Term weights are calculated according to the *tf/idf* function. Term selection was not considered thus obtaining a set of 28,006 distinct features.

We evaluated our framework on the binary category ranking task induced by the original multi-label classification task, thus requiring rankings having target classes of the original multi-label problem on top. Five different well-known cost functions have been used. Let $\mathbf{x}$ be an instance having ranking label $y$. **IErr** is the cost function indicating a non-perfect ranking and corresponds to the identity mapping in Figure 1-(a). **DErr** is the cost defined as the number of relevant classes uncorrectly ranked by the algorithm and corresponds to the domination mapping in Figure 1-(b). **dErr** is the cost obtained counting the number of uncorrect rankings and corresponds to the disagreement mapping in Figure 1-(c). Other two well-known Information Retrieval (IR) based cost functions have been used. The **OneErr** cost function that is 1 whenever the top ranked class is not a relevant class and the average precision cost function, which is $\mathbf{AvgP} = \frac{1}{|y|} \sum_{r \in y} \frac{|\{r' \in y : \text{rank}(\mathbf{x},r') \leq \text{rank}(\mathbf{x},r)\}|}{\text{rank}(\mathbf{x},r)}$.

**Results**   The model evaluation has been performed by comparing three different label mappings for KPLM and the baseline MMP algorithm [4], a variant of the Perceptron algorithm for ranking problems, with respect to the above-mentioned ranking losses. We used the configuration which gave the best results in the experiments reported in [4]. KPLM has been implemented setting $s = m$ and the standard basis vectors $\mathbf{e}_r \in \mathbb{R}^m$ as codes associated to the classes. A linear kernel $k(\mathbf{x}, \mathbf{y}) = (\langle \mathbf{x}, \mathbf{y} \rangle + 1)$ was used. Model selection for the KPLM has been performed by means of a 5-fold cross validation for different values

of the parameter $C$. The optimal parameters have been chosen as the ones minimizing the mean of the values of the loss (the one used for training) over the different folders. In Table 1 we report the obtained results. It is clear that KPLM definitely outperforms the MMP method. This is probably due to the use of margins in KPLM. Moreover, using identity and domination mappings seems to lead to models that outperform the ones obtained by using the disagreement mapping. Interestingly, this also happens when comparing with respect to its own corresponding cost. This can be due to a looser approximation (as a sum of approximations) of the true cost function. The same trend was confirmed by another set of experiments on artificial datasets that we are not able to report here due to space limitations.

| Method | IErr % | DErr % | dErr % | OneErr % | AvgP % |
|---|---|---|---|---|---|
| MMP | 5.07 | 4.92 | 0.89 | 4.28 | 97.49 |
| KPLM ($\mathcal{G}_I$) | **3.77** | 3.66 | 0.55 | **3.10** | **98.25** |
| KPLM ($\mathcal{G}_D$) | 3.81 | **3.59** | **0.54** | 3.14 | 98.24 |
| KPLM ($\mathcal{G}_d$) | 4.12 | 4.13 | 0.66 | 3.58 | 97.99 |

Table 1: Comparisons of ranking performance for different methods using different loss functions according to different evaluation metrics. Best results are shown in bold.

## 5 Conclusions and Future Work

We have presented a common framework for the analysis of general multiclass problems and proposed a kernel-based method as an instance of this setting which has shown very good results on a binary category ranking task. Promising directions of research, that we are currently pursuing, include experimenting with coding optimization and considering to extend the current setting to on-line learning, interdependent labels (e.g. hierarchical or any other structured classification), ordinal regression problems, and classification with costs.

## References

[1] F. Aiolli. *Large Margin Multiclass Learning: Models and Algorithms*. PhD thesis, Dept. of Computer Science, University of Pisa, 2004. http://www.di.unipi.it/˜ aiolli/thesis.ps.

[2] F. Aiolli and A. Sperduti. Multi-prototype support vector machine. In *Proceedings of International Joint Conference of Artificial Intelligence (IJCAI)*, 2003.

[3] K. Crammer and Y. Singer. On the learnability and design of output codes for multiclass problems. In *Proceedings of the Thirteenth Annual Conference on Computational Learning Theory*, pages 35–46, 2000.

[4] K. Crammer and Y. Singer. A new family of online algorithms for category ranking. *Journal of Machine Learning Research*, 2003.

[5] O. Dekel, C.D. Manning, and Y. Singer. Log-linear models for label ranking. In *Advances in Neural Information Processing Systems*, 2003.

[6] R.O. Duda, P.E. Hart, and D.G. Stork. *Pattern Classification*, chapter 5, page 266. Wiley, 2001.

[7] S. Har Peled, D. Roth, and D. Zimak. Constraint classification: A new approach to multiclass classification. In *Proceedings of the 13th International Conference on Algorithmic Learning Theory (ALT-02)*, 2002.

[8] G. Rätsch, A. Smola, and S. Mika. Adapting codes and embeddings for polychotomies. In *Advances in Neural Information Processing Systems*, 2002.

[9] V. Vapnik. *Statistical Learning Theory*. Wiley, New York, NY, 1998.

[10] J. Weston and C. Watkins. Multiclass support vector machines. In M. Verleysen, editor, *Proceedings of ESANN99*. D. Facto Press, 1999.

[11] T. Zhang and F.J. Oles. Text categorization based on regularized linear classification methods. *Information Retrieval*, 1(4):5–31, 2001.
